# SIMPLIFYING NEURAL NETS BY DISCOVERING FLAT MINIMA

Sepp Hochreiter*                    Jürgen Schmidhuber[†]
Fakultät für Informatik, H2
Technische Universität München
80290 München, Germany

## Abstract

We present a new algorithm for finding low complexity networks with high generalization capability. The algorithm searches for large connected regions of so-called "flat" minima of the error function. In the weight-space environment of a "flat" minimum, the error remains approximately constant. Using an MDL-based argument, flat minima can be shown to correspond to low expected overfitting. Although our algorithm requires the computation of second order derivatives, it has backprop's order of complexity. Experiments with feedforward and recurrent nets are described. In an application to stock market prediction, the method outperforms conventional backprop, weight decay, and "optimal brain surgeon".

## 1  INTRODUCTION

Previous algorithms for finding low complexity networks with high generalization capability are based on significant prior assumptions. They can be broadly classified as follows: *(1) Assumptions about the prior weight distribution.* Hinton and van Camp [3] and Williams [17] assume that pushing the posterior distribution (after learning) close to the prior leads to "good" generalization. Weight decay can be derived e.g. from Gaussian priors. Nowlan and Hinton [10] assume that networks with many similar weights generated by Gaussian mixtures are "better" a priori. MacKay's priors [6] are implicit in additional penalty terms, which embody the

*hochreit@informatik.tu-muenchen.de

[†]schmidhu@informatik.tu-muenchen.de

assumptions made. *(2) Prior assumptions about how theoretical results on early stopping and network complexity carry over to practical applications.* Examples are methods based on validation sets (see [8]), Vapnik's "structural risk minimization" [1] [14], and the methods of Holden [5] and Wang et al. [15]. Our approach requires less prior assumptions than most other approaches (see appendix A.1).

**Basic idea of flat minima search.** Our algorithm finds a large region in weight space with the property that each weight vector from that region has *similar* small error. Such regions are called "flat minima". To get an intuitive feeling for why "flat" minima are interesting, consider this (see also Wolpert [18]): a "sharp" minimum corresponds to weights which have to be specified with high precision. A "flat" minimum corresponds to weights many of which can be given with low precision. In the terminology of the theory of minimum description length (MDL), fewer bits of information are required to pick a "flat" minimum (corresponding to a "simple" or low complexity-network). The MDL principle suggests that low network complexity corresponds to high generalization performance (see e.g. [4, 13]). Unlike Hinton and van Camp's method [3] (see appendix A.3), our approach does not depend on explicitly choosing a "good" prior.

Our algorithm finds "flat" minima by searching for weights that minimize both training error and weight precision. This requires the computation of the Hessian. However, by using Pearlmutter's and Møller's efficient second order method [11, 7], we obtain the same order of complexity as with conventional backprop. *Automatically, the method effectively reduces numbers of units, weigths, and input lines, as well as the sensitivity of outputs with respect to remaining weights and units.* Excellent experimental generalization results will be reported in section 4.

# 2    TASK / ARCHITECTURE / BOXES

**Generalization task.** The task is to approximate an unknown relation $\bar{D} \subset X \times Z$ between a set of inputs $X \subset R^N$ and a set of outputs $Z \subset R^K$. $\bar{D}$ is taken to be a function. A relation $D$ is obtained from $\bar{D}$ by adding noise to the outputs. All training information is given by a finite relation $D_0 \subset D$. $D_0$ is called the *training set*. The $p$th element of $D_0$ is denoted by an input/target pair $(x_p, d_p)$.

**Architecture.** For simplicity, we will focus on a standard feedforward net (but in the experiments, we will use recurrent nets as well). The net has $N$ input units, $K$ output units, $W$ weights, and differentiable activation functions. It maps input vectors $x_p \in R^N$ to output vectors $o_p \in R^K$. The weight from unit $j$ to $i$ is denoted by $w_{ij}$. The $W$-dimensional weight vector is denoted by $w$.

**Training error.** Mean squared error $E_q(w, D_0) := \frac{1}{|D_0|} \sum_{(x_p, d_p) \in D_0} \| d_p - o_p \|^2$ is used, where $\| . \|$ denotes the Euclidian norm, and $|.|$ denotes the cardinality of a set. To define regions in weight space with the property that each weight vector from that region has "*similar* small error", we introduce the tolerable error $E_{tol}$, a positive constant. "Small" error is defined as being smaller than $E_{tol}$. $E_q(w, D_0) > E_{tol}$ implies "underfitting".

**Boxes.** Each weight $w$ satisfying $E_q(w, D_0) \leq E_{tol}$ defines an "acceptable minimum". **We are interested in large regions of connected acceptable minima.**

**Such regions are called flat minima. They are associated with low expected generalization error** (see [4]). To simplify the algorithm for finding large connected regions (see below), we do not consider maximal connected regions but focus on so-called *"bozes"* within regions: for each acceptable minimum $w$, its *box* $M_w$ in weight space is a $W$-dimensional hypercuboid with center $w$. For simplicity, each edge of the box is taken to be parallel to one weight axis. Half the length of the box edge in direction of the axis corresponding to weight $w_{ij}$ is denoted by $\Delta w_{ij}$, which is the maximal (positive) value such that for all $i, j$, all positive $\kappa_{ij} \leq \Delta w_{ij}$ can be added to or subtracted from the corresponding component of $w$ simultaneously without violating $E_q(., D_0) \leq E_{tol}$ ($\Delta w_{ij}$ gives the precision of $w_{ij}$). $M_w$'s *box volume* is defined by $\Delta w := 2^W \prod_{i,j} \Delta w_{ij}$.

## 3   THE ALGORITHM

The algorithm is designed to find a $w$ defining a box $M_w$ with maximal box volume $\Delta w$. This is equivalent to finding a box $M_w$ with minimal $\tilde{B}(w, D_0) := -\log(\Delta w / 2^W) = \sum_{i,j} -\log \Delta w_{ij}$. Note the relationship to MDL ($\tilde{B}$ is the number of bits required to describe the weights). In appendix A.2, we derive the following algorithm. It minimizes $E(w, D_0) = E_q(w, D_0) + \lambda B(w, D_0)$, where

$$B = \frac{1}{2} \left( -W \log \epsilon + \sum_{i,j} \log \sum_k (\frac{\partial o^k}{\partial w_{ij}})^2 + W \log \sum_k \left( \sum_{i,j} \frac{|\frac{\partial o^k}{\partial w_{ij}}|}{\sqrt{\sum_k (\frac{\partial o^k}{\partial w_{ij}})^2}} \right)^2 \right). \quad (1)$$

Here $o^k$ is the activation of the $k$th output unit, $\epsilon$ is a constant, and $\lambda$ is a positive variable ensuring either $E_q(w, D_0) \leq E_{tol}$, or ensuring an expected decrease of $E_q(., D_0)$ during learning (see [16] for adjusting $\lambda$).

$E(w, D_0)$ is minimized by gradient descent. To minimize $B(w, D_0)$, we compute

$$\frac{\partial B(w, D_0)}{\partial w_{uv}} = \sum_{k,i,j} \frac{\partial B(w, D_0)}{\partial (\frac{\partial o^k}{\partial w_{ij}})} \frac{\partial^2 o^k}{\partial w_{ij} \partial w_{uv}} \text{ for all } u, v . \quad (2)$$

It can be shown (see [4]) that by using Pearlmutter's and Møller's efficient second order method [11, 7], the gradient of $B(w, D_0)$ can be computed in $O(W)$ time (see details in [4]). **Therefore, our algorithm has the same order of complexity as standard backprop.**

## 4   EXPERIMENTAL RESULTS (see [4] for details)

**EXPERIMENT 1 – noisy classification.** The first experiment is taken from Pearlmutter and Rosenfeld [12]. The task is to decide whether the $x$-coordinate of a point in 2-dimensional space exceeds zero (class 1) or does not (class 2). Noisy training examples are generated as follows: data points are obtained from a Gaussian with zero mean and stdev 1.0, bounded in the interval $[-3.0, 3.0]$. The data points are misclassified with a probability of 0.05. Final input data is obtained by adding a zero mean Gaussian with stdev 0.15 to the data points. In a test with 2,000,000 data points, it was found that the procedure above leads to 9.27 per cent

| | Backprop | | New approach | | | Backprop | | New approach | |
|---|---|---|---|---|---|---|---|---|---|
| | MSE | dto | MSE | dto | | MSE | dto | MSE | dto |
| 1 | 0.220 | 1.35 | 0.193 | 0.00 | 6 | 0.219 | 1.24 | 0.187 | 0.04 |
| 2 | 0.223 | 1.16 | 0.189 | 0.09 | 7 | 0.215 | 1.14 | 0.187 | 0.07 |
| 3 | 0.222 | 1.37 | 0.186 | 0.13 | 8 | 0.214 | 1.10 | 0.185 | 0.01 |
| 4 | 0.213 | 1.18 | 0.181 | 0.01 | 9 | 0.218 | 1.21 | 0.190 | 0.09 |
| 5 | 0.222 | 1.24 | 0.195 | 0.25 | 10 | 0.214 | 1.21 | 0.188 | 0.07 |

Table 1: *10 comparisons of conventional backprop (BP) and our new method (FMS). The second row (labeled "MSE") shows mean squared error on the test set. The third row ("dto") shows the difference between the fraction (in per cent) of misclassifications and the optimal fraction (9.27). The remaining rows provide the analoguous information for the new approach, which clearly outperforms backprop.*

misclassified data. *No* method will misclassify less than 9.27 per cent, due to the inherent noise in the data. The training set is based on 200 fixed data points. The test set is based on 120,000 data points.

**Results.** 10 conventional backprop (BP) nets were tested against 10 equally initialized networks based on our new method ("flat minima search", FMS). *After 1,000 epochs, the weights of our nets essentially stopped changing (automatic "early stopping"), while backprop kept changing weights to learn the outliers in the data set and overfit.* In the end, our approach left a single hidden unit $h$ with a maximal weight of 30.0 or −30.0 from the x-axis input. Unlike with backprop, the other hidden units were effectively pruned away (outputs near zero). So was the y-axis input (zero weight to $h$). It can be shown that this corresponds to an "optimal" net with minimal numbers of units and weights. Table 1 illustrates the superior performance of our approach.

**EXPERIMENT 2 – recurrent nets.** The method works for continually running fully recurrent nets as well. At every time step, a recurrent net with sigmoid activations in $[0, 1]$ sees an input vector from a stream of randomly chosen input vectors from the set $\{(0,0),(0,1),(1,0),(1,1)\}$. The task is to switch on the first output unit whenever an input $(1, 0)$ had occurred two time steps ago, and to switch on the second output unit without delay in response to any input $(0, 1)$. The task can be solved by a single hidden unit.

**Results.** With conventional recurrent net algorithms, after training, both hidden units were used to store the input vector. Not so with our new approach. We trained 20 networks. All of them learned perfect solutions. Like with weight decay, most weights to the output decayed to zero. But *unlike* with weight decay, **strong inhibitory** connections (-30.0) switched off one of the hidden units, effectively pruning it away.

**EXPERIMENT 3 – stock market prediction.** We predict the DAX (German stock market index) based on fundamental (experiments 3.1 and 3.2) and technical (experiment 3.3) indicators. We use strictly layered feedforward nets with sigmoid units active in [-1,1], and the following performance measures:

*Confidence:* output $o > \alpha \rightarrow$ positive tendency, $o < -\alpha \rightarrow$ negative tendency.
*Performance:* Sum of confidently, incorrectly predicted DAX changes is subtracted

from sum of confidently, correctly predicted ones. The result is divided by the sum of absolute changes.

EXPERIMENT 3.1: Fundamental inputs: (a) German interest rate ( *"Umlaufsrendite"*), (b) industrial production divided by money supply, (c) business sentiments ( *"IFO Geschäftsklimaindex"*). 24 training examples, 68 test examples, quarterly prediction, confidence: $\alpha = 0.0/0.6/0.9$, architecture: (3-8-1).

EXPERIMENT 3.2: Fundamental inputs: (a), (b), (c) as in exp. 3.1, (d) dividend rate, (e) foreign orders in manufacturing industry. 228 training examples, 100 test examples, monthly prediction, confidence: $\alpha = 0.0/0.6/0.8$, architecture: (5-8-1).

EXPERIMENT 3.3: Technical inputs: (a) 8 most recent DAX-changes, (b) DAX, (c) change of 24-week relative strength index ("RSI"), (d) difference of "5 week statistic", (e) "MACD" (difference of exponentially weighted 6 week and 24 week DAX). 320 training examples, 100 test examples, weekly predictions, confidence: $\alpha = 0.0/0.2/0.4$, architecture: (12-9-1).

The following methods are tested: (1) Conventional backprop (BP), (2) optimal brain surgeon (OBS [2]), (3) weight decay (WD [16]), (4) flat minima search (FMS).

**Results.** Our method clearly outperforms the other methods. FMS is up to 63 per cent better than the best competitor (see [4] for details).

# APPENDIX – THEORETICAL JUSTIFICATION

## A.1. OVERFITTING ERROR

In analogy to [15] and [1], we decompose the generalization error into an "overfitting" error and an "underfitting" error. There is no significant underfitting error (corresponding to Vapnik's empirical risk) if $E_q(w, D_0) \leq E_{tol}$. Some thought is required, however, to define the "overfitting" error. We do this in a novel way. Since we do not know the relation $D$, we cannot know $p(\alpha \mid D)$, the "optimal" posterior weight distribution we would obtain by training the net on $D$ ($\rightarrow$ "sure thing hypothesis"). But, for theoretical purposes, suppose we *did* know $p(\alpha \mid D)$. Then we could use $p(\alpha \mid D)$ to initialize weights before learning the training set $D_0$. Using the Kullback-Leibler distance, we measure the information (due to noise) conveyed by $D_0$, but not by $D$. In conjunction with the initialization above, this provides the conceptual setting for defining an overfitting error measure. But, the initialization does not really matter, because it does not heavily influence the posterior (see [4]).

The overfitting error is the Kullback-Leibler distance of the posteriors: $E_o(D, D_0) = \int p(\alpha \mid D_0) \log (p(\alpha \mid D_0)/p(\alpha \mid D)) \, d\alpha$. $E_o(D, D_0)$ is the expectation of $\log (p(\alpha \mid D_0)/p(\alpha \mid D))$ (the expected difference of the minimal description of $\alpha$ with respect to $D$ and $D_0$, after learning $D_0$). Now we measure the **expected overfitting error relative to** $M_w$ (see section 2) by computing the expectation of $\log (p(\alpha \mid D_0)/p(\alpha \mid D))$ in the range $M_w$:

$$E_{ro}(w) = \beta \left( \int_{M_w} p_{M_w}(\alpha \mid D_0) E_q(\alpha, D) d\alpha - \bar{E}_q(D_0, M_w) \right) . \qquad (3)$$

Here $p_{M_w}(\alpha \mid D_0) := p(\alpha \mid D_0)/ \int_{M_w} p(\tilde{\alpha} \mid D_0) d\tilde{\alpha}$ is the posterior of $D_0$ scaled to obtain a distribution within $M_w$, and $\bar{E}_q(D_0, M_w) := \int_{M_w} p_{M_w}(\alpha \mid D_0) E_q(\alpha, D_0) d\alpha$ is the mean error in $M_w$ with respect to $D_0$.

Clearly, we would like to pick $w$ such that $E_{ro}(w)$ is minimized. Towards this purpose, we need two additional *prior assumptions*, which are actually implicit in most previous approaches (which make additional stronger assumptions, see section 1): *(1) "Closeness assumption"*: Every minimum of $E_q(., D_0)$ is "close" to a maximum of $p(\alpha|D)$ (see formal definition in [4]). Intuitively, "closeness" ensures that $D_0$ can indeed tell us something about $D$, such that training on $D_0$ may indeed reduce the error on $D$. *(2) "Flatness assumption"*: The peaks of $p(\alpha|D)$'s maxima are not sharp. This MDL-like assumption holds if not *all* weights have to be known exactly to model $D$. It ensures that there are regions with low error on $D$.

## A.2. HOW TO FLATTEN THE NETWORK OUTPUT

To find nets with flat outputs, two conditions will be defined to specify $B(w, D_0)$ (see section 3). The first condition ensures flatness. The second condition enforces "equal flatness" in all weight space directions. In both cases, linear approximations will be made (to be justified in [4]). We are looking for weights (causing tolerable error) that can be perturbed without causing significant output changes. Perturbing the weights $w$ by $\delta w$ (with components $\delta w_{ij}$), we obtain $ED(w, \delta w) := \sum_k (o^k(w + \delta w) - o^k(w))^2$, where $o^k(w)$ expresses $o^k$'s dependence on $w$ (in what follows, however, $w$ often will be suppressed for convenience). Linear approximation (justified in [4]) gives us **"Flatness Condition 1"**:

$$ED(w, \delta w) \approx \sum_k (\sum_{i,j} \frac{\partial o^k}{\partial w_{ij}} \delta w_{ij})^2 \leq \sum_k (\sum_{i,j} |\frac{\partial o^k}{\partial w_{ij}}| |\delta w_{ij}|)^2 \leq \epsilon , \qquad (4)$$

where $\epsilon > 0$ defines tolerable output changes within a box and is small enough to allow for linear approximation (it does not appear in $B(w, D_0)$'s gradient, see section 3).

Many $M_w$ satisfy flatness condition 1. To select a particular, very flat $M_w$, the following **"Flatness Condition 2"** uses up degrees of freedom left by (4):

$$\forall i, j, u, v : (\delta w_{ij})^2 \sum_k (\frac{\partial o^k}{\partial w_{ij}})^2 = (\delta w_{uv})^2 \sum_k (\frac{\partial o^k}{\partial w_{uv}})^2 . \qquad (5)$$

Flatness Condition 2 enforces equal "directed errors" $ED_{ij}(w, \delta w_{ij}) = \sum_k (o^k(w_{ij} + \delta w_{ij}) - o^k(w_{ij}))^2 \approx \sum_k (\frac{\partial o^k}{\partial w_{ij}} \delta w_{ij})^2$, where $o^k(w_{ij})$ has the obvious meaning. *It can be shown* (see [4]) *that with given box volume, we* **need** *flatness condition 2 to minimize the expected description length of the box center.* Flatness condition 2 influences the algorithm as follows: (1) The algorithm prefers to increase the $\delta w_{ij}$'s of weights which currently are not important to generate the target output. (2) The algorithm enforces equal sensitivity of all output units with respect to the weights. Hence, the algorithm tends to group hidden units according to their relevance for groups of output units. Flatness condition 2 is essential: flatness condition 1 by itself corresponds to nothing more but first order derivative reduction (ordinary sensitivity reduction, e.g. [9]). Linear approximation is justified by the choice of $\epsilon$ in equation (4).

We first solve equation (5) for $|\delta w_{ij}| = |\delta w_{uv}| \left( \sqrt{\sum_k \left(\frac{\partial o^k}{\partial w_{uv}}\right)^2} / \sqrt{\sum_k \left(\frac{\partial o^k}{\partial w_{ij}}\right)^2} \right)$

(fixing $u, v$ for all $i, j$). Then we insert $|\delta w_{ij}|$ into equation (4) (replacing the second "$\leq$" in (4) by "$=$"). This gives us an equation for the $|\delta w_{ij}|$ (which depend on $w$, but this is notationally suppressed):

$$|\delta w_{ij}| = \sqrt{\epsilon} / \left( \sqrt{\sum_k (\frac{\partial o^k}{\partial w_{ij}})^2} \sqrt{\sum_k \left( \sum_{i,j} \frac{|\frac{\partial o^k}{\partial w_{ij}}|}{\sqrt{\sum_k (\frac{\partial o^k}{\partial w_{ij}})^2}} \right)^2} \right). \quad (6)$$

The $|\delta w_{ij}|$ approximate the $\Delta w_{ij}$ from section 2. Thus, $\tilde{B}(w, D_0)$ (see section 3) can be approximated by $B(w, D_0) := \sum_{i,j} -\log |\delta w_{ij}|$. This immediately leads to the algorithm given by equation (1).

How can this approximation be justified? **The learning process itself enforces its validity (see justification in [4]).** Initially, the conditions above are valid only in a very small environment of an "initial" acceptable minimum. But during search for new acceptable minima with more associated box volume, the corresponding environments are enlarged, which implies that the absolute values of the entries in the Hessian decrease. It can be shown (see [4]) that the algorithm tends to suppress the following values: (1) unit activations, (2) first order activation derivatives, (3) the sum of all contributions of an arbitary unit activation to the net output. Since weights, inputs, activation functions, and their first and second order derivatives are bounded, it can be shown (see [4]) that the entries in the Hessian decrease where the corresponding $|\delta w_{ij}|$ increase.

## A.3. RELATION TO HINTON AND VAN CAMP

Hinton and van Camp [3] minimize the sum of two terms: the first is conventional error plus variance, the other is the distance $\int p(\alpha \mid D_0) \log (p(\alpha \mid D_0)/p(\alpha)) \, d\alpha$ between posterior $p(\alpha \mid D_0)$ and prior $p(\alpha)$. The problem is to choose a "good" prior. In contrast to their approach, our approach does not require a "good" prior given in advance. Furthermore, Hinton and van Camp have to compute variances of weights and units, which (in general) *cannot* be done using linear approximation. Intuitively speaking, their weight variances are related to our $\Delta w_{ij}$. Our approach, however, *does* justify linear approximation.

## References

[1] I. Guyon, V. Vapnik, B. Boser, L. Bottou, and S. A. Solla. Structural risk minimization for character recognition. In J. E. Moody, S. J. Hanson, and R. P. Lippman, editors, *Advances in Neural Information Processing Systems 4*, pages 471–479. San Mateo, CA: Morgan Kaufmann, 1992.

[2] B. Hassibi and D. G. Stork. Second order derivatives for network pruning: Optimal brain surgeon. In J. D. Cowan S. J. Hanson and C. L. Giles, editors, *Advances in Neural Information Processing Systems 5*, pages 164–171. San Mateo, CA: Morgan Kaufmann, 1993.

[3] G. E. Hinton and D. van Camp. Keeping neural networks simple. In *Proceedings of the International Conference on Artificial Neural Networks, Amsterdam*, pages 11–18. Springer, 1993.

[4] S. Hochreiter and J. Schmidhuber. Flat minima search for discovering simple nets. Technical Report FKI-200-94, Fakultät für Informatik, Technische Universität München, 1994.

[5] S. B. Holden. *On the Theory of Generalization and Self-Structuring in Linearly Weighted Connectionist Networks.* PhD thesis, Cambridge University, Engineering Department, 1994.

[6] D. J. C. MacKay. A practical Bayesian framework for backprop networks. *Neural Computation*, 4:448–472, 1992.

[7] M. F. Møller. Exact calculation of the product of the Hessian matrix of feed-forward network error functions and a vector in O(N) time. Technical Report PB-432, Computer Science Department, Aarhus University, Denmark, 1993.

[8] J. E. Moody and J. Utans. Architecture selection strategies for neural networks: Application to corporate bond rating prediction. In A. N. Refenes, editor, *Neural Networks in the Capital Markets*. John Wiley & Sons, 1994.

[9] A. F. Murray and P. J. Edwards. Synaptic weight noise during MLP learning enhances fault-tolerance, generalisation and learning trajectory. In J. D. Cowan S. J. Hanson and C. L. Giles, editors, *Advances in Neural Information Processing Systems 5*, pages 491–498. San Mateo, CA: Morgan Kaufmann, 1993.

[10] S. J. Nowlan and G. E. Hinton. Simplifying neural networks by soft weight sharing. *Neural Computation*, 4:173–193, 1992.

[11] B. A. Pearlmutter. Fast exact multiplication by the Hessian. *Neural Computation*, 1994.

[12] B. A. Pearlmutter and R. Rosenfeld. Chaitin-Kolmogorov complexity and generalization in neural networks. In R. P. Lippmann, J. E. Moody, and D. S. Touretzky, editors, *Advances in Neural Information Processing Systems 3*, pages 925–931. San Mateo, CA: Morgan Kaufmann, 1991.

[13] J. H. Schmidhuber. Discovering problem solutions with low Kolmogorov complexity and high generalization capability. Technical Report FKI-194-94, Fakultät für Informatik, Technische Universität München, 1994.

[14] V. Vapnik. Principles of risk minimization for learning theory. In J. E. Moody, S. J. Hanson, and R. P. Lippman, editors, *Advances in Neural Information Processing Systems 4*, pages 831–838. San Mateo, CA: Morgan Kaufmann, 1992.

[15] C. Wang, S. S. Venkatesh, and J. S. Judd. Optimal stopping and effective machine complexity in learning. In J. D. Cowan, G. Tesauro, and J. Alspector, editors, *Advances in Neural Information Processing Systems 6*, pages 303–310. Morgan Kaufmann, San Mateo, CA, 1994.

[16] A. S. Weigend, D. E. Rumelhart, and B. A. Huberman. Generalization by weight-elimination with application to forecasting. In R. P. Lippmann, J. E. Moody, and D. S. Touretzky, editors, *Advances in Neural Information Processing Systems 3*, pages 875–882. San Mateo, CA: Morgan Kaufmann, 1991.

[17] P. M. Williams. Bayesian regularisation and pruning using a Laplace prior. Technical report, School of Cognitive and Computing Sciences, University of Sussex, Falmer, Brighton, 1994.

[18] D. H. Wolpert. Bayesian backpropagation over i-o functions rather than weights. In J. D. Cowan, G. Tesauro, and J. Alspector, editors, *Advances in Neural Information Processing Systems 6*, pages 200–207. San Mateo, CA: Morgan Kaufmann, 1994.